# Indian Buffet Processes with Power-law Behavior

**Yee Whye Teh and Dilan Görür**
Gatsby Computational Neuroscience Unit, UCL
17 Queen Square, London WC1N 3AR, United Kingdom
{ywteh,dilan}@gatsby.ucl.ac.uk

## Abstract

The Indian buffet process (IBP) is an exchangeable distribution over binary matrices used in Bayesian nonparametric featural models. In this paper we propose a three-parameter generalization of the IBP exhibiting power-law behavior. We achieve this by generalizing the beta process (the de Finetti measure of the IBP) to the *stable-beta process* and deriving the IBP corresponding to it. We find interesting relationships between the stable-beta process and the Pitman-Yor process (another stochastic process used in Bayesian nonparametric models with interesting power-law properties). We derive a stick-breaking construction for the stable-beta process, and find that our power-law IBP is a good model for word occurrences in document corpora.

## 1 Introduction

The Indian buffet process (IBP) is an infinitely exchangeable distribution over binary matrices with a finite number of rows and an unbounded number of columns [1, 2]. It has been proposed as a suitable prior for Bayesian nonparametric featural models, where each object (row) is modeled with a potentially unbounded number of features (columns). Applications of the IBP include Bayesian nonparametric models for ICA [3], choice modeling [4], similarity judgements modeling [5], dyadic data modeling [6] and causal inference [7].

In this paper we propose a three-parameter generalization of the IBP with power-law behavior. Using the usual analogy of customers entering an Indian buffet restaurant and sequentially choosing dishes from an infinitely long buffet counter, our generalization with parameters $\alpha > 0$, $c > -\sigma$ and $\sigma \in [0, 1)$ is simply as follows:

- Customer 1 tries $\mathrm{Poisson}(\alpha)$ dishes.
- Subsequently, customer $n + 1$:
    - tries dish $k$ with probability $\frac{m_k - \sigma}{n + c}$, for each dish that has previously been tried;
    - tries $\mathrm{Poisson}(\alpha \frac{\Gamma(1+c)\Gamma(n+c+\sigma)}{\Gamma(n+1+c)\Gamma(c+\sigma)})$ new dishes.

where $m_k$ is the number of previous customers who tried dish $k$. The dishes and the customers correspond to the columns and the rows of the binary matrix respectively, with an entry of the matrix being one if the corresponding customer tried the dish (and zero otherwise). The mass parameter $\alpha$ controls the total number of dishes tried by the customers, the concentration parameter $c$ controls the number of customers that will try each dish, and the stability exponent $\sigma$ controls the power-law behavior of the process. When $\sigma = 0$ the process does not exhibit power-law behavior and reduces to the usual two-parameter IBP [2].

Many naturally occurring phenomena exhibit power-law behavior, and it has been argued that using models that can capture this behavior can improve learning [8]. Recent examples where this has led to significant improvements include unsupervised morphology learning [8], language modeling [9]

and image segmentation [10]. These examples are all based on the Pitman-Yor process [11, 12, 13], a generalization of the Dirichlet process [14] with power-law properties. Our generalization of the IBP extends the ability to model power-law behavior to featural models, and we expect it to lead to a wealth of novel applications not previously well handled by the IBP.

The approach we take in this paper is to first define the underlying de Finetti measure, then to derive the conditional distributions of Bernoulli process observations with the de Finetti measure integrated out. This automatically ensures that the resulting power-law IBP is infinitely exchangeable. We call the de Finetti measure of the power-law IBP the *stable-beta process*. It is a novel generalization of the beta process [15] (which is the de Finetti measure of the normal two-parameter IBP [16]) with characteristics reminiscent of the stable process [17, 11] (in turn related to the Pitman-Yor process). We will see that the stable-beta process has a number of properties similar to the Pitman-Yor process.

In the following section we first give a brief description of completely random measures, a class of random measures which includes the stable-beta and the beta processes. In Section 3 we introduce the stable-beta process, a three parameter generalization of the beta process and derive the power-law IBP based on the stable-beta process. Based on the proposed model, in Section 4 we construct a model of word occurrences in a document corpus. We conclude with a discussion in Section 5.

## 2 Completely Random Measures

In this section we give a brief description of completely random measures [18]. Let $\Theta$ be a measure space with $\Omega$ its $\sigma$-algebra. A random variable whose values are measures on $(\Theta, \Omega)$ is referred to as a random measure. A completely random measure (CRM) $\mu$ over $(\Theta, \Omega)$ is a random measure such that $\mu(A) \perp\!\!\!\perp \mu(B)$ for all disjoint measurable subsets $A, B \in \Omega$. That is, the (random) masses assigned to disjoint subsets are independent. An important implication of this property is that the whole distribution over $\mu$ is determined (with usually satisfied technical assumptions) once the distributions of $\mu(A)$ are given for all $A \in \Omega$.

CRMs can always be decomposed into a sum of three independent parts: a (non-random) measure, an atomic measure with fixed atoms but random masses, and an atomic measure with random atoms and masses. CRMs in this paper will only contain the second and third components. In this case we can write $\mu$ in the form,

$$\mu = \sum_{k=1}^{N} u_k \delta_{\phi_k} + \sum_{l=1}^{M} v_l \delta_{\psi_l}, \tag{1}$$

where $u_k, v_l > 0$ are the random masses, $\phi_k \in \Theta$ are the fixed atoms, $\psi_l \in \Theta$ are the random atoms, and $N, M \in \mathbb{N} \cup \{\infty\}$. To describe $\mu$ fully it is sufficient to specify $N$ and $\{\phi_k\}$, and to describe the joint distribution over the random variables $\{u_k\}, \{v_l\}, \{\psi_l\}$ and $M$. Each $u_k$ has to be independent from everything else and has some distribution $F_k$. The random atoms and their weights $\{v_l, \psi_l\}$ are jointly drawn from a 2D Poisson process over $(0, \infty] \times \Theta$ with some nonatomic rate measure $\Lambda$ called the Lévy measure. The rate measure $\Lambda$ has to satisfy a number of technical properties; see [18, 19] for details. If $\int_\Theta \int_{(0,\infty]} \Lambda(du \times d\theta) = M^* < \infty$ then the number of random atoms $M$ in $\mu$ is Poisson distributed with mean $M^*$, otherwise there are an infinite number of random atoms. If $\mu$ is described by $\Lambda$ and $\{\phi_k, F_k\}_{k=1}^{N}$ as above, we write,

$$\mu \sim \mathrm{CRM}(\Lambda, \{\phi_k, F_k\}_{k=1}^{N}). \tag{2}$$

## 3 The Stable-beta Process

In this section we introduce a novel CRM called the stable-beta process (SBP). It has no fixed atoms while its Lévy measure is defined over $(0, 1) \times \Theta$:

$$\Lambda_0(du \times d\theta) = \alpha \frac{\Gamma(1+c)}{\Gamma(1-\sigma)\Gamma(c+\sigma)} u^{-\sigma-1}(1-u)^{c+\sigma-1} du H(d\theta) \tag{3}$$

where the parameters are: a mass parameter $\alpha > 0$, a concentration parameter $c > -\sigma$, a stability exponent $0 \le \sigma < 1$, and a smooth base distribution $H$. The mass parameter controls the overall mass of the process and the base distribution gives the distribution over the random atom locations.

The mean of the SBP can be shown to be $\mathbb{E}[\mu(A)] = \alpha H(A)$ for each $A \in \Omega$, while $\mathrm{var}(\mu(A)) = \alpha \frac{1-\sigma}{1+c} H(A)$. Thus the concentration parameter and the stability exponent both affect the variability of the SBP around its mean. The stability exponent also governs the power-law behavior of the SBP. When $\sigma = 0$ the SBP does not have power-law behavior and reduces to a normal two-parameter beta process [15, 16]. When $c = 1 - \sigma$ the stable-beta process describes the random atoms with masses $< 1$ in a stable process [17, 11]. The SBP is so named as it can be seen as a generalization of both the stable and the beta processes. Both the concentration parameter and the stability exponent can be generalized to functions over $\Theta$ though we will not deal with this generalization here.

### 3.1  Posterior Stable-beta Process

Consider the following hierarchical model:

$$\mu \sim \mathrm{CRM}(\Lambda_0, \{\}),$$
$$Z_i | \mu \sim \mathrm{BernoulliP}(\mu) \qquad\qquad \text{iid, for } i = 1, \dots, n. \qquad (4)$$

The random measure $\mu$ is a SBP with no fixed atoms and with Lévy measure (3), while $Z_i \sim$ BernoulliP$(\mu)$ is a Bernoulli process with mean $\mu$ [16]. This is also a CRM: in a small neighborhood $d\theta$ around $\theta \in \Theta$ it has a probability $\mu(d\theta)$ of having a unit mass atom in $d\theta$; otherwise it does not have an atom in $d\theta$. If $\mu$ has an atom at $\theta$ the probability of $Z_i$ having an atom at $\theta$ as well is $\mu(\{\theta\})$. If $\mu$ has a smooth component, say $\mu_0$, $Z_i$ will have random atoms drawn from a Poisson process with rate measure $\mu_0$. In typical applications to featural models the atoms in $Z_i$ give the features associated with data item $i$, while the weights of the atoms in $\mu$ give the prior probabilities of the corresponding features occurring in a data item.

We are interested in both the posterior of $\mu$ given $Z_1, \dots, Z_n$, as well as the conditional distribution of $Z_{n+1} | Z_1, \dots, Z_n$ with $\mu$ marginalized out. Let $\theta_1^*, \dots, \theta_K^*$ be the $K$ unique atoms among $Z_1, \dots, Z_n$ with atom $\theta_k^*$ occurring $m_k$ times. Theorem 3.3 of [20] shows that the posterior of $\mu$ given $Z_1, \dots, Z_n$ is still a CRM, but now including fixed atoms given by $\theta_1^*, \dots, \theta_K^*$. Its updated Lévy measure and the distribution of the mass at each fixed atom $\theta_k^*$ are,

$$\mu | Z_1, \dots, Z_n \sim \mathrm{CRM}(\Lambda_n, \{\theta_k^*, F_{nk}\}_{k=1}^K), \qquad (5)$$

where

$$\Lambda_n(du \times d\theta) = \alpha \frac{\Gamma(1+c)}{\Gamma(1-\sigma)\Gamma(c+\sigma)} u^{-\sigma-1}(1-u)^{n+c+\sigma-1} du H(d\theta), \qquad (6a)$$

$$F_{nk}(du) = \frac{\Gamma(n+c)}{\Gamma(m_k-\sigma)\Gamma(n-m_k+c+\sigma)} u^{m_k-\sigma-1}(1-u)^{n-m_k+c+\sigma-1} du. \qquad (6b)$$

Intuitively, the posterior is obtained as follows. Firstly, the posterior of $\mu$ must be a CRM since both the prior of $\mu$ and the likelihood of each $Z_i | \mu$ factorize over disjoint subsets of $\Theta$. Secondly, $\mu$ must have fixed atoms at each $\theta_k^*$ since otherwise the probability that there will be atoms among $Z_1, \dots, Z_n$ at precisely $\theta_k^*$ is zero. The posterior mass at $\theta_k^*$ is obtained by multiplying a Bernoulli "likelihood" $u^{m_k}(1-u)^{n-m_k}$ (since there are $m_k$ occurrences of the atom $\theta_k^*$ among $Z_1, \dots, Z_n$) to the "prior" $\Lambda_0(du \times d\theta_k^*)$ in (3) and normalizing, giving us (6b). Finally, outside of these $K$ atoms there are no other atoms among $Z_1, \dots, Z_n$. We can think of this as $n$ observations of 0 among $n$ iid Bernoulli variables, so a "likelihood" of $(1-u)^n$ is multiplied into $\Lambda_0$ (without normalization), giving the updated Lévy measure in (6a).

Let us inspect the distributions (6) of the fixed and random atoms in the posterior $\mu$ in turn. The random mass at $\theta_k^*$ has a distribution $F_{nk}$ which is simply a beta distribution with parameters $(m_k - \sigma, n - m_k + c + \sigma)$. This differs from the usual beta process in the subtraction of $\sigma$ from $m_k$ and addition of $\sigma$ to $n - m_k + c$. This is reminiscent of the Pitman-Yor generalization to the Dirichlet process [11, 12, 13], where a discount parameter is subtracted from the number of customers seated around each table, and added to the chance of sitting at a new table. On the other hand, the Lévy measure of the random atoms of $\mu$ is still a Lévy measure corresponding to an SBP with updated parameters

$$\alpha' \leftarrow \alpha \frac{\Gamma(1+c)\Gamma(n+c+\sigma)}{\Gamma(n+1+c)\Gamma(c+\sigma)}, \qquad\qquad \sigma' \leftarrow \sigma$$
$$c' \leftarrow c + n, \qquad\qquad H' \leftarrow H. \qquad (7)$$

Note that the update depends only on $n$, not on $Z_1, \ldots, Z_n$. In summary, the posterior of $\mu$ is simply an independent sum of an SBP with updated parameters and of fixed atoms with beta distributed masses. Observe that the posterior $\mu$ is not itself a SBP. In other words, the SBP is not conjugate to Bernoulli process observations. This is different from the beta process and again reminiscent of Pitman-Yor processes, where the posterior is also a sum of a Pitman-Yor process with updated parameters and fixed atoms with random masses, but not a Pitman-Yor process [11]. Fortunately, the non-conjugacy of the SBP does not preclude efficient inference. In the next subsections we describe an Indian buffet process and a stick-breaking construction corresponding to the SBP. Efficient inference techniques based on both representations for the beta process can be straightforwardly generalized to the SBP [1, 16, 21].

### 3.2 The Stable-beta Indian Buffet Process

We can derive an Indian buffet process (IBP) corresponding to the SBP by deriving, for each $n$, the distribution of $Z_{n+1}$ conditioned on $Z_1, \ldots, Z_n$, with $\mu$ marginalized out. This derivation is straightforward and follows closely that for the beta process [16]. For each of the atoms $\theta_k^*$ the posterior of $\mu(\theta_k^*)$ given $Z_1, \ldots, Z_n$ is beta distributed with mean $\frac{m_k - \sigma}{n+c}$. Thus

$$p(Z_{n+1}(\theta_k^*) = 1 | Z_1, \ldots, Z_n) = \mathbb{E}[\mu(\theta_k^*) | Z_1, \ldots, Z_n] = \frac{m_k - \sigma}{n+c} \qquad (8)$$

Metaphorically speaking, customer $n+1$ tries dish $k$ with probability $\frac{m_k - \sigma}{n+c}$. Now for the random atoms. Let $\theta \in \Theta \backslash \{\theta_1^*, \ldots, \theta_K^*\}$. In a small neighborhood $d\theta$ around $\theta$, we have:

$$p(Z_{n+1}(d\theta) = 1 | Z_1, \ldots, Z_n) = \mathbb{E}[\mu(d\theta) | Z_1, \ldots, Z_n] = \int_0^1 u \Lambda_n(du \times d\theta)$$

$$= \int_0^1 u \alpha \frac{\Gamma(1+c)}{\Gamma(1-\sigma)\Gamma(c+\sigma)} u^{-1-\sigma}(1-u)^{n+c+\sigma-1} du H(d\theta)$$

$$= \alpha \frac{\Gamma(1+c)}{\Gamma(1-\sigma)\Gamma(c+\sigma)} H(d\theta) \int_0^1 u^{-\sigma}(1-u)^{n+c+\sigma-1} du$$

$$= \alpha \frac{\Gamma(1+c)\Gamma(n+c+\sigma)}{\Gamma(n+1+c)\Gamma(c+\sigma)} H(d\theta) \qquad (9)$$

Since $Z_{n+1}$ is completely random and $H$ is smooth, the above shows that on $\Theta \backslash \{\theta_1^*, \ldots, \theta_K^*\}$ $Z_{n+1}$ is simply a Poisson process with rate measure $\alpha \frac{\Gamma(1+c)\Gamma(n+c+\sigma)}{\Gamma(n+1+c)\Gamma(c+\sigma)} H$. In particular, it will have $\text{Poisson}(\alpha \frac{\Gamma(1+c)\Gamma(n+c+\sigma)}{\Gamma(n+1+c)\Gamma(c+\sigma)})$ new atoms, each independently and identically distributed according to $H$. In the IBP metaphor, this corresponds to customer $n+1$ trying new dishes, with each dish associated with a new draw from $H$. The resulting Indian buffet process is as described in the introduction. It is automatically infinitely exchangeable since it was derived from the conditional distributions of the hierarchical model (4).

Multiplying the conditional probabilities of each $Z_n$ given previous ones together, we get the joint probability of $Z_1, \ldots, Z_n$ with $\mu$ marginalized out:

$$p(Z_1, \ldots, Z_n) = \exp\left(-\alpha \sum_{i=1}^n \frac{\Gamma(1+c)\Gamma(i-1+c+\sigma)}{\Gamma(i+c)\Gamma(c+\sigma)}\right) \prod_{k=1}^K \frac{\Gamma(m_k-\sigma)\Gamma(n-m_k+c+\sigma)\Gamma(1+c)}{\Gamma(1-\sigma)\Gamma(c+\sigma)\Gamma(n+c)} \alpha h(\theta_k^*), \quad (10)$$

where there are $K$ atoms (dishes) $\theta_1^*, \ldots, \theta_K^*$ among $Z_1, \ldots, Z_n$ with atom $k$ appearing $m_k$ times, and $h$ is the density of $H$. (10) is to be contrasted with (4) in [1]. The $K_h!$ terms in [1] are absent as we have to distinguish among these $K_h$ dishes in assigning each of them a distinct atom (this also contributes the $h(\theta_k^*)$ terms). The fact that (10) is invariant to permuting the ordering among $Z_1, \ldots, Z_n$ also indicates the infinite exchangeability of the stable-beta IBP.

### 3.3 Stick-breaking constructions

In this section we describe stick-breaking constructions for the SBP generalizing those for the beta process. The first is based on the size-biased ordering of atoms induced by the IBP [16], while

the second is based on the inverse Lévy measure method [22], and produces a sequence of random atoms of strictly decreasing masses [21].

The size-biased construction is straightforward: we use the IBP to generate the atoms (dishes) in the SBP; each time a dish is newly generated the atom is drawn from $H$ and its mass from $F_{nk}$. This leads to the following procedure:

$$\text{for } n = 1, 2, \ldots: \qquad J_n \sim \text{Poisson}(\alpha \tfrac{\Gamma(1+c)\Gamma(n-1+c+\sigma)}{\Gamma(n+c)\Gamma(c+\sigma)}),$$

$$\text{for } k = 1, \ldots, J_n: \qquad v_{nk} \sim \text{Beta}(1 - \sigma, n - 1 + c + \sigma), \qquad \psi_{nk} \sim H, \qquad (11)$$

$$\mu = \sum_{n=1}^{\infty} \sum_{k=1}^{J_n} v_{nk} \delta_{\psi_{nk}}.$$

The inverse Lévy measure is a general method of generating from a Poisson process with non-uniform rate measure. It essentially transforms the Poisson process into one with uniform rate, generates a sample, and transforms the sample back. This method is more involved for the SBP because the inverse transform has no analytically tractable form. The Lévy measure $\Lambda_0$ of the SBP factorizes into a product $\Lambda_0(du \times d\theta) = L(du)H(d\theta)$ of a $\sigma$-finite measure $L(du) = \alpha \frac{\Gamma(1+c)}{\Gamma(1-\sigma)\Gamma(c+\sigma)} u^{-\sigma-1}(1-u)^{c+\sigma-1} du$ over $(0,1)$ and a probability measure $H$ over $\Theta$. This implies that we can generate a sample $\{v_l, \psi_l\}_{l=1}^{\infty}$ of the random atoms of $\mu$ and their masses by first sampling the masses $\{v_l\}_{l=1}^{\infty} \sim \text{PoissonP}(L)$ from a Poisson process on $(0,1)$ with rate measure $L$, and associating each $v_l$ with an iid draw $\psi_l \sim H$ [19]. Now consider the mapping $T : (0,1) \to (0,\infty)$ given by

$$T(u) = \int_u^1 L(du) = \int_u^1 \alpha \frac{\Gamma(1+c)}{\Gamma(1-\sigma)\Gamma(c+\sigma)} u^{-\sigma-1}(1-u)^{c+\sigma-1} du. \qquad (12)$$

$T$ is bijective and monotonically decreasing. The Mapping Theorem for Poisson processes [19] shows that $\{v_l\}_{l=1}^{\infty} \sim \text{PoissonP}(L)$ if and only if $\{T(v_l)\}_{l=1}^{\infty} \sim \text{PoissonP}(\mathcal{L})$ where $\mathcal{L}$ is Lebesgue measure on $(0,\infty)$. A sample $\{t_l\}_{l=1}^{\infty} \sim \text{PoissonP}(\mathcal{L})$ can be easily drawn by letting $e_l \sim \text{Exponential}(1)$ and setting $t_l = \sum_{i=1}^{l} e_i$ for all $l$. Transforming back with $v_l = T^{-1}(t_l)$, we have $\{v_l\}_{l=1}^{\infty} \sim \text{PoissonP}(L)$. As $t_1, t_2, \ldots$ is an increasing sequence and $T$ is decreasing, $v_1, v_2, \ldots$ is a decreasing sequence of masses. Deriving the density of $v_l$ given $v_{l-1}$, we get:

$$p(v_l|v_{l-1}) = \left|\tfrac{dt_l}{dv_l}\right| p(t_l|t_{l-1}) = \alpha \tfrac{\Gamma(1+c)}{\Gamma(1-\sigma)\Gamma(c+\sigma)} v_l^{-\sigma-1}(1-v_l)^{c+\sigma-1} \exp\left\{-\int_{v_l}^{v_{l-1}} L(du)\right\}. \quad (13)$$

In general these densities do not simplify and we have to resort to solving for $T^{-1}(t_l)$ numerically. There are two cases for which they do simplify. For $c = 1$, $\sigma = 0$, the density function reduces to $p(v_l|v_{l-1}) = \alpha v_l^{\alpha-1}/v_{l-1}^{\alpha}$, leading to the stick-breaking construction of the single parameter IBP [21]. In the stable process case when $c = 1 - \sigma$ and $\sigma \neq 0$, the density of $v_l$ simplifies to:

$$p(v_l \mid v_{l-1}) = \alpha \tfrac{\Gamma(2-\sigma)}{\Gamma(1-\sigma)\Gamma(1)} v_l^{-\sigma-1} \times \exp\left\{-\int_{v_l}^{v_{l-1}} \alpha \tfrac{\Gamma(2-\sigma)}{\Gamma(1-\sigma)\Gamma(1)} u^{-\sigma-1} du\right\}$$

$$= \alpha(1-\sigma) v_l^{-\sigma-1} \exp\left\{-\tfrac{\alpha(1-\sigma)}{\sigma}(v_l^{-\sigma} - v_{l-1}^{-\sigma})\right\}. \qquad (14)$$

Doing a change of values to $y_l = v_l^{-\sigma}$, we get:

$$p(y_l|y_{l-1}) = \alpha \tfrac{1-\sigma}{\sigma} \exp\left\{-\alpha \tfrac{1-\sigma}{\sigma}(y_l - y_{l-1})\right\}. \qquad (15)$$

That is, each $y_l$ is exponentially distributed with rate $\alpha \frac{1-\sigma}{\sigma}$ and offset by $y_{l-1}$. For general values of the parameters we do not have an analytic stick breaking form. However note that the weights generated using this method are still going to be strictly decreasing.

### 3.4 Power-law Properties

The SBP has a number of appealing power-law properties. In this section we shall assume $\sigma > 0$ since the case $\sigma = 0$ reduces the SBP to the usual beta process with less interesting power-law properties. Derivations are given in the appendix.

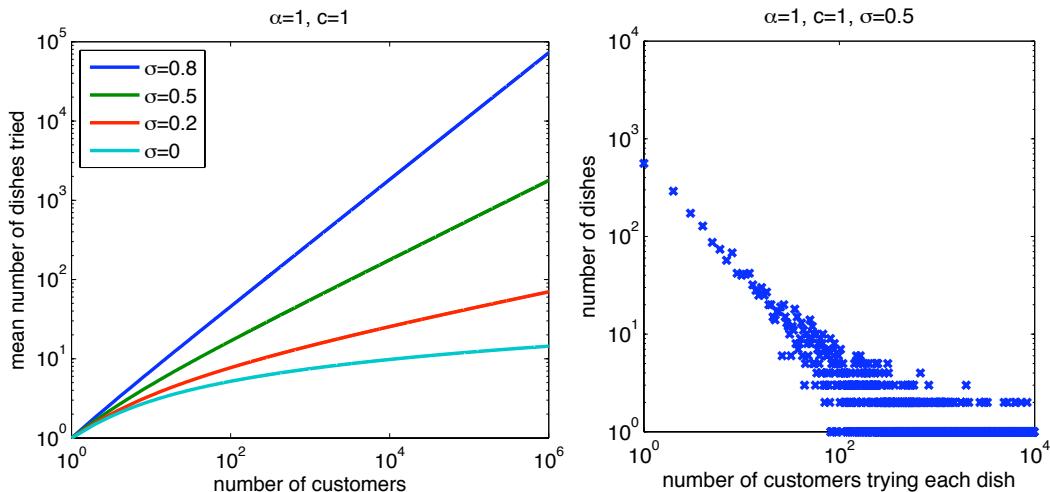

Figure 1: Power-law properties of the stable-beta Indian buffet process.

Firstly, the total number of dishes tried by $n$ customers is $\mathbb{O}(n^\sigma)$. The left panel of Figure 1 shows this for varying $\sigma$. Secondly, the number of customers trying each dish follows a Zipf's law [23]. This is shown in the right panel of Figure 1, which plots the number of dishes $K_m$ versus the number of customers $m$ trying each dish (that is, $K_m$ is the number of dishes $k$ for which $m_k = m$). Asymptotically we can show that the proportion of dishes tried by $m$ customers is $\mathbb{O}(m^{-1-\sigma})$. Note that these power-laws are similar to those observed for Pitman-Yor processes. One aspect of the SBP which is not power-law is the number of dishes each customer tries. This is simply $\text{Poisson}(\alpha)$ distributed. It seems difficult obtain power-law behavior in this aspect within a CRM framework, because of the fundamental role played by the Poisson process.

## 4   Word Occurrence Models with Stable-beta Processes

In this section we use the SBP as a model for word occurrences in document corpora. Let $n$ be the number of documents in a corpus. Let $Z_i(\{\theta\}) = 1$ if word type $\theta$ occurs in document $i$ and 0 otherwise, and let $\mu(\{\theta\})$ be the occurrence probability of word type $\theta$ among the documents in the corpus. We use the hierarchical model (4) with a SBP prior[1] on $\mu$ and with each document modeled as a conditionally independent Bernoulli process draw. The joint distribution over the word occurrences $Z_1, \ldots, Z_n$, with $\mu$ integrated out, is given by the IBP joint probability (10).

We applied the word occurrence model to the 20newsgroups dataset. Following [16], we modeled the training documents in each of the 20 newsgroups as a separate corpus with a separate SBP. We use the popularity of each word type across all 20 newsgroups as the base distribution[2]: for each word type $\theta$ let $n_\theta$ be the number of documents containing $\theta$ and let $H(\{\theta\}) \propto n_\theta$.

In the first experiment we compared the SBP to the beta process by fitting the parameters $\alpha, c$ and $\sigma$ of both models to each newsgroup by maximum likelihood (in beta process case $\sigma$ is fixed at 0) . We expect the SBP to perform better as it is better able to capture the power-law statistics of the document corpora (see Figure 2). The ML values of the parameters across classes did not vary much, taking values $\alpha = 142.6 \pm 40.0, c = 4.1 \pm 0.9$ and $\sigma = 0.47 \pm 0.1$. In comparison, the parameters values obtained by the beta process are $\alpha = 147.3 \pm 41.4$ and $c = 25.9 \pm 8.4$. Note that the estimated values for $c$ are significantly larger than for the SBP to allow the beta process to model the fact that many words occur in a small number of documents (a consequence of the power-law

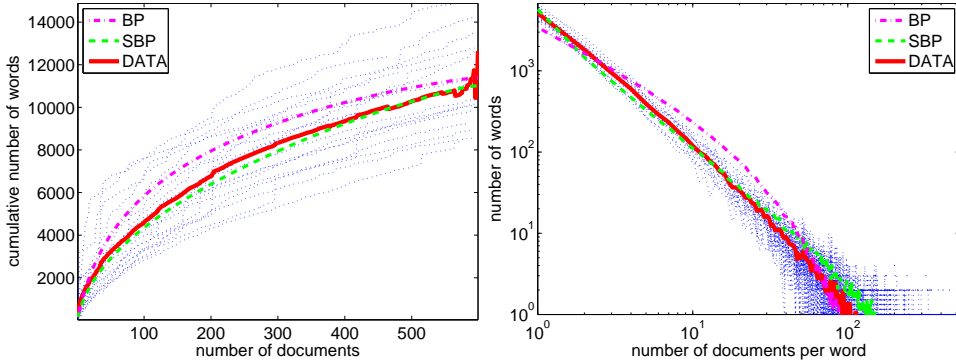

Figure 2: Power-law properties of the 20newsgroups dataset. The faint dashed lines are the distributions of words in the documents in each class, the solid curve is the mean of these lines. The dashed lines are the means of the word distributions generated by the ML parameters for the beta process (pink) and the SBP (green).

Table 1: Classification performance of SBP and beta process (BP). The $j$th column (denoted $1{:}j$) shows the cumulative rank $j$ classification accuracy of the test documents. The three numbers after the models are the percentages of training, validation and test sets respectively.

| assigned to classes: | 1 | 1:2 | 1:3 | 1:4 | 1:5 |
|---|---|---|---|---|---|
| BP - 20/20/60 | 78.7($\pm$0.5) | 87.4($\pm$0.2) | 91.3($\pm$0.2) | 95.1($\pm$0.2) | 96.2($\pm$0.2) |
| SBP - 20/20/60 | 79.9($\pm$0.5) | 87.6($\pm$0.1) | 91.5($\pm$0.2) | 93.7($\pm$0.2) | 95.1($\pm$0.2) |
| BP - 60/20/20 | 85.5($\pm$0.6) | 91.6($\pm$0.3) | 94.2($\pm$0.3) | 95.6($\pm$0.4) | 96.6($\pm$0.3) |
| SBP - 60/20/20 | 85.5($\pm$0.4) | 91.9($\pm$0.4) | 94.4($\pm$0.2) | 95.6($\pm$0.3) | 96.6($\pm$0.3) |

statistics of word occurrences; see Figure 2). We also plotted the characteristics of data simulated from the models using the estimated ML parameters. The SBP has a much better fit than the beta process to the power-law properties of the corpora.

In the second experiment we tested the two models on categorizing test documents into one of the 20 newsgroups. Since this is a discriminative task, we optimized the parameters in both models to maximize the cumulative ranked classification performance. The rank $j$ classification performance is defined to be the percentage of documents where the true label is among the top $j$ predicted classes (as determined by the IBP conditional probabilities of the documents under each of the 20 newsgroup classes). As the cost function is not differentiable, we did a grid search over the parameter space, using 20 values of $\alpha, c$ and $\sigma$ each, and found the parameters maximizing the objective function on a validation set separate from the test set. To see the effect of sample size on model performance we tried splitting the documents in each newsgroup into 20% training, 20% validation and 60% test sets, and into 60% training, 20% validation and 20% test sets. We repeated the experiment five times with different random splits of the dataset. The ranked classification rates are shown in Table 1. Figure 3 shows that the SBP model has generally higher classification performances than the beta process.

## 5 Discussion

We have introduced a novel stochastic process called the stable-beta process. The stable-beta process is a generalization of the beta process, and can be used in nonparametric Bayesian featural models with an unbounded number of features. As opposed to the beta process, the stable-beta process has a number of appealing power-law properties. We developed both an Indian buffet process and a stick-breaking construction for the stable-beta process and applied it to modeling word occurrences in document corpora. We expect the stable-beta process to find uses modeling a range of natural phenomena with power-law properties.

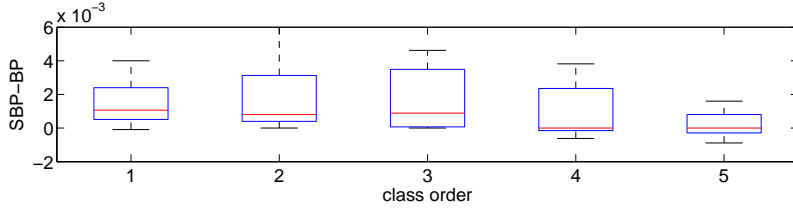

Figure 3: Differences between the classification rates of the SBP and the beta process. The performance of the SBP was consistently higher than that of the beta process for each of the five runs.

We derived the stable-beta process as a completely random measure with Lévy measure (3). It would be interesting and illuminating to try to derive it as an infinite limit of finite models, however we were not able to do so in our initial attempts. A related question is whether there is a natural definition of the stable-beta process for non-smooth base distributions. Until this is resolved in the positive, we are not able to define hierarchical stable-beta processes generalizing the hierarchical beta processes [16].

Another avenue of research we are currently pursuing is in deriving better stick-breaking constructions for the stable-beta process. The current construction requires inverting the integral (12), which is expensive as it requires an iterative method which evaluates the integral numerically within each iteration.

### Acknowledgement

We thank the Gatsby Charitable Foundation for funding, Romain Thibaux, Peter Latham and Tom Griffiths for interesting discussions, and the anonymous reviewers for help and feedback.

## A    Derivation of Power-law Properties

We will make large $n$ and $K$ assumptions here, and make use of Stirling's approximation $\Gamma(n+1) \approx \sqrt{2\pi n}(n/e)^n$, which is accurate in the larger $n$ regime. The expected number of dishes is,

$$\mathbb{E}[K] = \sum_{i=1}^{n} \alpha \frac{\Gamma(1+c)\Gamma(n+c+\sigma)}{\Gamma(n+1+c)\Gamma(c+\sigma)} \in \mathbb{O}\left(\sum_{i=1}^{n} \frac{\sqrt{2\pi(i+c+\sigma-1)}((i+c+\sigma-1)/e)^{i+c+\sigma-1}}{\sqrt{2\pi(i+c)}((i+c)/e)^{i+c}}\right)$$

$$=\mathbb{O}\left(\sum_{i=1}^{n} e^{-\sigma+1}(1 + \tfrac{\sigma-1}{i+c})^{i+c}(i+c+\sigma-1)^{\sigma-1}\right) = \mathbb{O}\left(\sum_{i=1}^{n} e^{-\sigma+1}e^{\sigma-1}i^{\sigma-1}\right) = \mathbb{O}(n^{\sigma}). \quad (16)$$

We are interested in the joint distribution of the statistics $(K_1, \ldots, K_n)$, where $K_m$ is the number of dishes tried by exactly $m$ customers and where there are a total of $n$ customers in the restaurant. As there are $\frac{K!}{\prod_{m=1}^{n} K_m!} \prod_{m=1}^{n} \left(\frac{n!}{m!(n-m)!}\right)^{K_m}$ configurations of the IBP with the same statistics $(K_1, \ldots, K_n)$, we have (ignoring constant terms and collecting terms in (10) with $m_k = m$),

$$p(K_1, \ldots, K_n|n) \propto \frac{K!}{\prod_{m=1}^{n} K_m!} \prod_{m=1}^{n} \left(\frac{n!}{m!(n-m)!} \frac{\Gamma(m-\sigma)\Gamma(n-m+c+\sigma)\Gamma(1+c)}{\Gamma(1-\sigma)\Gamma(c+\sigma)\Gamma(n+c)}\right)^{K_m}. \quad (17)$$

Conditioning on $K = \sum_{m=1}^{n} K_m$ as well, we see that $(K_1, \ldots, K_n)$ is multinomial with the probability of a dish having $m$ customers being proportional to the term in large parentheses. For large $m$ (and even larger $n$), this probability simplifies to,

$$\mathbb{O}(\tfrac{\Gamma(m-\sigma)}{\Gamma(m+1)}) = \mathbb{O}\left(\frac{\sqrt{2\pi(m-1-\sigma)}((m-1-\sigma)/e)^{m-1-\sigma}}{\sqrt{2\pi m}(m/e)^m}\right) = \mathbb{O}\left(m^{-1-\sigma}\right). \quad (18)$$

## Footnotes

[1]Words are discrete objects. To get a smooth base distribution we imagine appending each word type with a $U[0, 1]$ variate. This does not affect the modelling that follows.

[2]The appropriate technique, as proposed by [16], would be to use a hierarchical SBP to tie the word occurrence probabilities across the newsgroups. However due to difficulties dealing with atomic base distributions we cannot define a hierarchical SBP easily (see discussion).

# References

[1] T. L. Griffiths and Z. Ghahramani. Infinite latent feature models and the Indian buffet process. In *Advances in Neural Information Processing Systems*, volume 18, 2006.

[2] Z. Ghahramani, T. L. Griffiths, and P. Sollich. Bayesian nonparametric latent feature models (with discussion and rejoinder). In *Bayesian Statistics*, volume 8, 2007.

[3] D. Knowles and Z. Ghahramani. Infinite sparse factor analysis and infinite independent components analysis. In *International Conference on Independent Component Analysis and Signal Separation*, volume 7 of *Lecture Notes in Computer Science*. Springer, 2007.

[4] D. Görür, F. Jäkel, and C. E. Rasmussen. A choice model with infinitely many latent features. In *Proceedings of the International Conference on Machine Learning*, volume 23, 2006.

[5] D. J. Navarro and T. L. Griffiths. Latent features in similarity judgment: A nonparametric Bayesian approach. *Neural Computation*, in press 2008.

[6] E. Meeds, Z. Ghahramani, R. M. Neal, and S. T. Roweis. Modeling dyadic data with binary latent factors. In *Advances in Neural Information Processing Systems*, volume 19, 2007.

[7] F. Wood, T. L. Griffiths, and Z. Ghahramani. A non-parametric Bayesian method for inferring hidden causes. In *Proceedings of the Conference on Uncertainty in Artificial Intelligence*, volume 22, 2006.

[8] S. Goldwater, T.L. Griffiths, and M. Johnson. Interpolating between types and tokens by estimating power-law generators. In *Advances in Neural Information Processing Systems*, volume 18, 2006.

[9] Y. W. Teh. A hierarchical Bayesian language model based on Pitman-Yor processes. In *Proceedings of the 21st International Conference on Computational Linguistics and 44th Annual Meeting of the Association for Computational Linguistics*, pages 985–992, 2006.

[10] E. Sudderth and M. I. Jordan. Shared segmentation of natural scenes using dependent Pitman-Yor processes. In *Advances in Neural Information Processing Systems*, volume 21, 2009.

[11] M. Perman, J. Pitman, and M. Yor. Size-biased sampling of Poisson point processes and excursions. *Probability Theory and Related Fields*, 92(1):21–39, 1992.

[12] J. Pitman and M. Yor. The two-parameter Poisson-Dirichlet distribution derived from a stable subordinator. *Annals of Probability*, 25:855–900, 1997.

[13] H. Ishwaran and L. F. James. Gibbs sampling methods for stick-breaking priors. *Journal of the American Statistical Association*, 96(453):161–173, 2001.

[14] T. S. Ferguson. A Bayesian analysis of some nonparametric problems. *Annals of Statistics*, 1(2):209–230, 1973.

[15] N. L. Hjort. Nonparametric Bayes estimators based on beta processes in models for life history data. *Annals of Statistics*, 18(3):1259–1294, 1990.

[16] R. Thibaux and M. I. Jordan. Hierarchical beta processes and the Indian buffet process. In *Proceedings of the International Workshop on Artificial Intelligence and Statistics*, volume 11, pages 564–571, 2007.

[17] M. Perman. *Random Discrete Distributions Derived from Subordinators*. PhD thesis, Department of Statistics, University of California at Berkeley, 1990.

[18] J. F. C. Kingman. Completely random measures. *Pacific Journal of Mathematics*, 21(1):59–78, 1967.

[19] J. F. C. Kingman. *Poisson Processes*. Oxford University Press, 1993.

[20] Y. Kim. Nonparametric Bayesian estimators for counting processes. *Annals of Statistics*, 27(2):562–588, 1999.

[21] Y. W. Teh, D. Görür, and Z. Ghahramani. Stick-breaking construction for the Indian buffet process. In *Proceedings of the International Conference on Artificial Intelligence and Statistics*, volume 11, 2007.

[22] R. L. Wolpert and K. Ickstadt. Simulations of lévy random fields. In *Practical Nonparametric and Semiparametric Bayesian Statistics*, pages 227–242. Springer-Verlag, 1998.

[23] G. Zipf. *Selective Studies and the Principle of Relative Frequency in Language*. Harvard University Press, Cambridge, MA, 1932.

